# The Performance of Convex Set Projection Based Neural Networks

Robert J. Marks II, Les E. Atlas, Seho Oh and James A. Ritcey

Interactive Systems Design Lab, FT-10
University of Washington, Seattle, Wa 98195.

## ABSTRACT

We consider a class of neural networks whose performance can be analyzed and geometrically visualized in a signal space environment. Alternating projection neural networks (APNN's) perform by alternately projecting between two or more constraint sets. Criteria for desired and unique convergence are easily established. The network can be configured in either a homogeneous or layered form. The number of patterns that can be stored in the network is on the order of the number of input and hidden neurons. If the output neurons can take on only one of two states, then the trained layered APNN can be easily configured to converge in one iteration. More generally, convergence is at an exponential rate. Convergence can be improved by the use of sigmoid type nonlinearities, network relaxation and/or increasing the number of neurons in the hidden layer. The manner in which the network responds to data for which it was not specifically trained (i.e. how it *generalizes*) can be directly evaluated analytically.

## 1. INTRODUCTION

In this paper, we depart from the performance analysis techniques normally applied to neural networks. Instead, a signal space approach is used to gain new insights via ease of analysis and geometrical interpretation. Building on a foundation laid elsewhere[1-3], we demonstrate that alternating projecting neural network's (APNN's) formulated from such a viewpoint can be configured in layered form or homogeneously.

Significantly, APNN's have advantages over other neural network architectures. For example,

(a) APNN's perform by alternatingly projecting between two or more constraint sets. Criteria can be established for proper iterative convergence for both synchronous and asynchronous operation. This is in contrast to the more conventional technique of formulation of an energy metric for the neural networks, establishing a lower energy bound and showing that the energy reduces each iteration[4-7]. Such procedures generally do not address the accuracy of the final solution. In order to assure that such networks arrive at the desired globally minimum energy, computationaly lengthly procedures such as simulated annealing are used[8-10]. For synchronous networks, steady state oscillation can occur between two states of the same energy[11]

(b) Homogeneous neural networks such as Hopfield's content addressable memory[4,12-14] do not scale well, i.e. the capacity

of Hopfield's neural networks less than doubles when the number of neurons is doubled [15-16]. Also, the capacity of previously proposed layered neural networks[17,18] is not well understood. The capacity of the *layered* APNN's, on the other hand, is roughly equal to the number of input and hidden neurons[19].

(c) The speed of backward error propagation learning [17-18] can be painfully slow. Layered APNN's, on the other hand, can be trained on only one pass through the training data[2]. If the network memory does not saturate, new data can easily be learned without repeating previous data. Neither is the effectiveness of recall of previous data diminished. Unlike layered back propagation neural networks, the APNN recalls by iteration. Under certain important applications, however, the APNN will recall in one iteration.

(d) The manner in which layered APNN's *generalizes* to data for which it was not trained can be analyzed straightforwardly.

The outline of this paper is as follows. After establishing the dynamics of the APNN in the next section, sufficient criteria for proper convergence are given. The convergence dynamics of the APNN are explored. Wise use of nonlinearities, e.g. the sigmoidal type nonlinearities[2], improve the network's performance. Establishing a hidden layer of neurons whose states are a nonlinear function of the input neurons' states is shown to increase the network's capacity and the network's convergence rate as well. The manner in which the networks respond to data outside of the training set is also addressed.

## 2. THE ALTERNATING PROJECTION NEURAL NETWORK

In this section, we established the notation for the APNN. Nonlinear modificiations to the network made to impose certain performance attributes are considered later.

Consider a set of N continuous level linearly independent library vectors (or patterns) of length $L > N$: $\{ \vec{f}_n \mid 0 \leq n \leq N \}$. We form the library matrix $\underline{F} = [ \vec{f}_1 \mid \vec{f}_{2} \mid ... \mid \vec{f}_{N} ]$ and the neural network interconnect matrix[a] $\underline{T} = \underline{F} (\underline{F}^T \underline{F})^{-1} \underline{F}^T$ where the superscript $T$ denotes transposition. We divide the L neurons into two sets: one in which the states are known and the remainder in which the states are unknown. This partition may change from application to application. Let $s_k(M)$ be the state of the $k^{th}$ node at time M. If the $k^{th}$ node falls into the known category, its state is *clamped* to the known value (i.e. $s_k(M) = \ell_k$ where $\vec{\ell}$ is some library vector). The states of the remaining *floating* neurons are equal to the sum of the inputs into the node. That is, $s_k(M) = i_k$, where

$$i_k = \sum_{p=1}^{L} t_{pk} s_p \qquad (1)$$

---

[a] The interconnect matrix is better trained iteratively[2]. To include a new library vector $\vec{f}$, the interconnects are updated as $\underline{T} + (\vec{\epsilon}\vec{\epsilon}^T) / (\vec{\epsilon}^T \vec{\epsilon})$ where $\vec{\epsilon} = (\underline{I} - \underline{T})\vec{f}$.

If all neurons change state simultaneously (i.e. $s_p = s_p(M-1)$), then the net is said to operate synchronously. If only one neuron changes state at a time, the network is operating asynchronously.

Let P be the number of clamped neurons. We have proven[1] that the neural states converge strongly to the extrapolated library vector if the first P rows of $\underline{F}$ (denoted $\underline{F}_P$) form a matrix of full column rank. That is, no column of $\underline{F}_P$ can be expressed as a linear combination of those remaining. By strong convergence[b], we mean $\lim_{M \to \infty} \| \vec{s}(M) - \vec{\ell} \| = 0$ where $\| \vec{x} \|^2 = \vec{x}^T \vec{x}$.

Lastly, note that subsumed in the criterion that $\underline{F}_P$ be full rank is the condition that the number of library vectors not exceed the number of known neural states ($P \geq N$). Techniques to bypass this restriction by using hidden neurons are discussed in section 5.

Partition Notation: Without loss of generality, we will assume that neurons 1 through P are clamped and the remaining neurons are floating. We adopt the vector partitioning notation

$$\vec{i} = \begin{bmatrix} \vec{i}_P \\ \vec{i}_Q \end{bmatrix}$$

where $\vec{i}_P$ is the P-tuple of the first P elements of $\vec{i}$ and $\vec{i}_Q$ is a vector of the remaining $Q = L-P$. We can thus write, for example, $\underline{F}_P = [\ \vec{f}_1^P\ |\vec{f}_2^P\ |...|\vec{f}_N^P\ ]$. Using this partition notation, we can define the neural clamping operator by:

$$\eta\, \vec{i} = \begin{bmatrix} \vec{\ell}^P \\ \vec{i}_Q \end{bmatrix}$$

Thus, the first P elements of $\vec{i}$ are clamped to $\vec{\ell}^P$. The remaining Q nodes "float".

Partition notation for the interconnect matrix will also prove useful. Define

$$\underline{T} = \begin{bmatrix} \underline{T}_2 & \underline{T}_1 \\ \hline \underline{T}_3 & \underline{T}_4 \end{bmatrix}$$

where $\underline{T}_2$ is a P by P and $\underline{T}_4$ a Q by Q matrix.

### 3. STEADY STATE CONVERGENCE PROOFS

For purposes of later reference, we address convergence of the network for synchronous operation. Asynchronous operation is addressed in reference 2. For proper convergence, both cases require that $\underline{F}_P$ be full rank. For synchronous operation, the network iteration in (1) followed by clamping can be written as:

$$\vec{s}(M+1) = \eta\ \underline{T}\ \vec{s}(M) \tag{2}$$

As is illustrated in[1-3], this operation can easily be visualized in an L dimensional signal space.

---

b  The referenced convergence proofs prove strong convergence in an infinite dimensional Hilbert space. In a discrete finite dimensional space, both strong and weak convergence imply uniform convergence[19,20], i.e. $\vec{s}(M) \to \vec{\ell}$ as $M \to \infty$.

For a given partition with P clamped neurons, (2) can be written in partitioned form as

$$\left[\begin{array}{c} \vec{f}^P \\ \hline \vec{s}^Q \ (M+1) \end{array}\right] = \eta \left[\begin{array}{c|c} \underline{T}_2 & \underline{T}_1 \\ \hline \underline{T}_3 & \underline{T}_4 \end{array}\right] \left[\begin{array}{c} \vec{f}^P \\ \hline \vec{s}^Q \ (M) \end{array}\right] \qquad (3)$$

The states of the P clamped neurons are not affected by their input sum. Thus, there is no contribution to the iteration by $\underline{T}_1$ and $\underline{T}_2$. We can equivalently write (3) as

$$\vec{s}^Q \ (M+1) = \underline{T}_3 \vec{f}^P + \underline{T}_4 \vec{s}^Q \ (M) \qquad (4)$$

We show in that if $\underline{F}_P$ is full rank, then the spectral radius (magnitude of the maximum eigenvalue) of $\underline{T}_4$ is strictly less than one[19]. It follows that the steady state solution of (4) is:

$$\vec{f}^Q = ( \underline{I} - \underline{T}_4 )^{-1} \ \underline{T}_3 \ \vec{f}^P \qquad (5)$$

where, since $\underline{F}_P$ is full rank, we have made use of our claim that

$$\vec{s}^Q \ (\infty) = \vec{f}^Q \qquad (6)$$

## 4. CONVERGENCE DYNAMICS

In this section, we explore different convergence dynamics of the APNN when $\underline{F}_P$ is full column rank. If the library matrix displays certain orthogonality characteristics, or if there is a single output (floating) neuron, convergence can be achieved in a single iteration. More generally, convergence is at an exponential rate. Two techniques are presented to improve convergence. The first is standard relaxation. Use of nonlinear convex constraint at each neuron is discussed elsewhere[2,19].

One Step Convergence: There are at least two important cases where the APNN converges other than uniformly in one iteration. Both require that the output be bipolar (±1). Convergence is in one step in the sense that

$$\vec{f}^Q = \underline{sign} \ \vec{s}^Q \ (1) \qquad (7)$$

where the vector operation sign takes the sign of each element of the vector on which it operates.

CASE 1: If there is a single output neuron, then, from (4),(5) and (6), $s^Q(1) = (1 - t_{LL}) \ell^Q$ . Since the eigenvalue of the (scalar) matrix, $\underline{T}_4 = t_{LL}$ lies between zero and one[19], we conclude that $1 - t_{LL} > 0$. Thus, if $\ell^Q$ is restricted to ±1, (7) follows immediately. A technique to extend this result to an arbitrary number of output neurons in a layered network is discussed in section 7.

CASE 2: For certain library matrices, the APNN can also display one step convergence. We showed that if the columns of $\underline{F}$ are orthogonal and the columns of $\underline{F}_P$ are also orthogonal, then one synchronous iteration results in floating states proportional to the steady

state values[19]. Specifically, for the floating neurons,

$$\vec{s}^Q(1) = \frac{\|\vec{z}^P\|^2}{\|\vec{z}\|^2}\vec{z}^Q \qquad (8)$$

An important special case of (8) is when the elements of $\underline{F}$ are all $\pm 1$ and orthogonal. If each element were chosen by a 50-50 coin flip, for example, we would expect (in the statistical sense) that this would be the case.

Exponential Convergence: More generally, the convergence rate of the APNN is exponential and is a function of the eigenstructure of $\underline{T}_4$. Let $\{\vec{p}_r \mid 1 \le r \le Q\}$ denote the eigenvectors of $\underline{T}_4$ and $\{\lambda_r\}$ the corresponding eigenvalues. Define $\underline{P} = [\vec{p}_1 | \vec{p}_2 | \ldots | \vec{p}_Q]$ and the diagonal matrix $\underline{\Lambda}_4$ such that diag $\underline{\Lambda}_4 = [\lambda_1 \ \lambda_2 \ \ldots \lambda_Q]^T$. Then we can write $\underline{T}_4 = \underline{P} \ \underline{\Lambda}_4 \ \underline{P}^T$. Define $\vec{x}(M) = \underline{P}^T \vec{s}(M)$. Since $\underline{P} \ \underline{P}^T = \underline{I}$, it follows from the difference equation in (4) that $\vec{x}(M+1) = \underline{P}^T \underline{T}_4 \underline{P} \ \underline{P}^T \vec{s}(M) + \underline{P}^T \underline{T}_3 \vec{z}^P = \underline{\Lambda}_4 \vec{x}(M) + \vec{g}$ where $\vec{g} = \underline{P}^T \underline{T}_3 \vec{z}^P$. The solution to this difference equation is

$$x_k(M) = \sum_{r=0}^{M} \lambda_k^r \ g_k = [1 - \lambda_k^{M+1}](1 - \lambda_k)^{-1} \ g_k \qquad (9)$$

Since the spectral radius of $\underline{T}_4$ is less than one[19], $\lambda_k^M \to 0$ as $M \to \infty$. Our steady state result is thus $x_k(\infty) = (1 - \lambda_k)^{-1} g_k$. Equation (9) can therefore be written as $x_k(M) = [1 - \lambda_k^{M+1}] x_k(\infty)$. The equivalent of a "time constant" in this exponential convergence is $1/\ell n(1/|\lambda_k|)$. The speed of convergence is thus dictated by the spectral radius of $\underline{T}_4$. As we have shown[19] later, adding neurons in a hidden layer in an APNN can significiantly reduce this spectral radius and thus improve the convergence rate.

Relaxation: Both the projection and clamping operations can be relaxed to alter the network's convergence without affecting its steady state[20-21]. For the interconnects, we choose an appropriate value of the relaxation parameter $\theta$ in the interval $(0,2)$ and redefine the interconnect matrix as $\underline{T}^\theta = \theta \underline{T} + (1 - \theta)\underline{I}$ or equivalently,

$$t_{nm}^\theta = \begin{cases} \theta(t_{nn} - 1) + 1 & ; \ n = m \\ \theta \ t_{nm} & ; \ n \ne m \end{cases}$$

To see the effect of such relaxation on convergence, we need simply examine the resulting eigenvalues. If $\underline{T}_4$ has eigenvalues $\{\lambda_r\}$, then $\underline{T}_4^\theta$ has eigenvalues $\lambda_r^\theta = 1 + \theta(\lambda_r - 1)$. A wise choice of $\theta$ reduces the spectral radius of $\underline{T}_4^\theta$ with respect to that of $\underline{T}_4$, and thus decreases the time constant of the network's convergence.

Any of the operators projecting onto convex sets can be relaxed without affecting steady state convergence[19-20]. These include the $\eta$ operator[2] and the sigmoid-type neural operator that projects onto a box. Choice of stationary relaxation parameters without numerical and/or empirical study of each specific case, however, generally remains more of an art than a science.

## 5. LAYERED APNN'S

The networks thus far considered are homogeneous in the sense that any neuron can be clamped or floating. If the partition is such that the same set of neurons always provides the network stimulus and the remainder respond, then the networks can be simplified. Clamped neurons, for example, ignore the states of the other neurons. The corresponding interconnects can then be deleted from the neural network architecture. When the neurons are so partitioned, we will refer the APNN as *layered*.

In this section, we explore various aspects of the layered APNN and in particular, the use of a so called hidden layer of neurons to increase the storage capacity of the network. An alternate architecture for a homogeneous APNN that require only Q neurons has been reported by Marks[2].

Hidden Layers: In its generic form, the APNN cannot perform a simple exclusive or (XOR). Indeed, failure to perform this same operation was a nail in the coffin of the perceptron[22]. Rumelhart et. al.[17-18] revived the perceptron by adding additional layers of neurons. Although doing so allowed nonlinear discrimination, the iterative training of such networks can be painfully slow. With the addition of a hidden layer, the APNN likewise generalizes. In contrast, the APNN can be trained by looking at each data vector only once[1].

Although neural networks will not likely be used for performing XOR's, their use in explaining the role of hidden neurons is quite instructive. The library matrix for the XOR is

$$\underline{F} = \begin{bmatrix} 0 & 0 & 1 & 1 \\ 0 & 1 & 0 & 1 \\ 0 & 1 & 1 & 0 \end{bmatrix}$$

The first two rows of $\underline{F}$ do not form a matrix of full column rank. Our approach is to augment $\underline{F}_p$ with two more rows such that the resulting matrix is full rank. Most any *nonlinear* combination of the first two rows will in general increase the matrix rank. Such a procedure, for example, is used in $\Phi$-classifiers[23]. Possible nonlinear operations include multiplication, a logical "AND" and running a weighted sum of the clamped neural states through a memoryless nonlinearity such as a sigmoid. This latter alteration is particularly well suited to neural architectures.

To illustrate with the exclusive or (XOR), a new hidden neural state is set equal to the exponentiation of the sum of the first two rows. A second hidden neurons will be assigned a value equal to the cosine of the sum of the first two neural states multiplied by $\pi/2$. (The choice of nonlinearities here is arbitrary.) The augmented library matrix is

$$\underline{F}_+ = \begin{bmatrix} 0 & 0 & 1 & 1 \\ 0 & 1 & 0 & 1 \\ \hline 1 & e & e & e^2 \\ 1 & 0 & 0 & -1 \\ \hline 0 & 1 & 1 & 0 \end{bmatrix}$$

In either the training or look-up mode, the states of the hidden neurons are clamped indirectly as a result of clamping the input neurons.

The playback architecture for this network is shown in Fig.1. The interconnect values for the dashed lines are unity. The remaining interconnects are from the projection matrix formed from $\underline{F}_+$.

Geometrical Interpretation : In lower dimensions, the effects of hidden neurons can be nicely illustrated geometrically. Consider the library matrix

$$\underline{F} = \left[ \begin{array}{cc} 1/2 & 1 \\ 1 & 1/2 \end{array} \right]$$

Clearly $\underline{F}_p = [1/2 \ 1]$. Let the neurons in the hidden layer be determined by the nonlineariy $x^2$ where x denotes the elements in the first row of $\underline{F}$. Then

$$\underline{F}_+ = \left[ \begin{array}{c|c} \vec{\ell}_1^+ & \vec{\ell}_2^+ \end{array} \right] = \left[ \begin{array}{cc} 1/2 & 1 \\ \hline 1/4 & 1 \\ \hline 1 & 1/2 \end{array} \right]$$

The corresponding geometry is shown in Fig.2 for x the input neuron, y the output and h the hidden neuron. The augmented library vectors are shown and a portion of the generated subspace is shown lightly shaded. The surface of $h = x^2$ resembles a cylindrical lens in three dimensions. Note that the linear variety corresponding to $x = 1/2$ intersects the cylindrical lens and subspace only at $\vec{\ell}_1^+$. Similarly, the $x = 1$ plane intersects the lens and subspace at $\vec{\ell}_2^+$. Thus, in both cases, clamping the input corresponding to the first element of one of the two library vectors uniquely determines the library vector.

Convergence Improvement: Use of additional neurons in the hidden layer will improve the convergence rate of the APNN[19]. Specifically, the spectral radius of the $\underline{T}_4$ matrix is decreased as additional neurons are added. The dominant time constant controlling convergence is thus decreased.

Capacity: Under the assumption that nonlinearities are chosen such that the augmented $\underline{F}_p$ matrix is of full rank, the number of vectors which can be stored in the layered APNN is equal to the sum of the number of neurons in the input and hidden layers. Note, then, that interconnects between the input and output neurons are not needed if there are a sufficiently large number of neurons in the hidden layer.

## 6. GENERALIZATION

We are assured that the APNN will converge to the desired result if a portion of a training vector is used to stimulate the network. What, however, will be the response if an initialization is used that is not in the training set or, in other words, how does the network *generalize* from the training set ?

To illustrate generalization, we return to the XOR problem. Let $s_5(M)$ denote the state of the output neuron at the $M^{th}$ (synchronous)

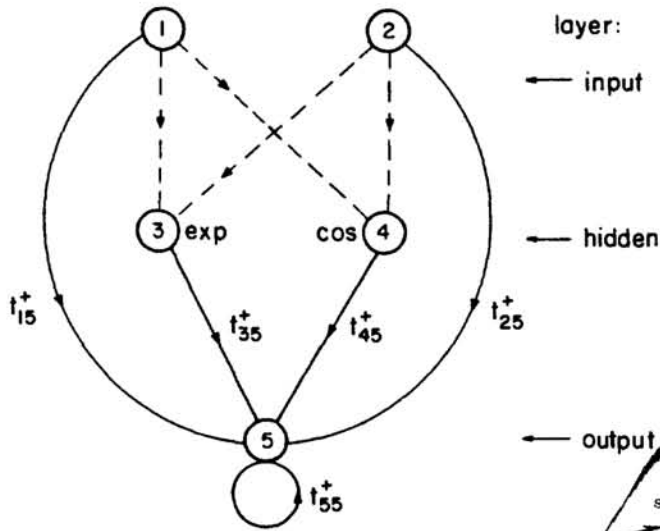

Figure 1. Illustration of a
layered APNN for performing
an XOR.

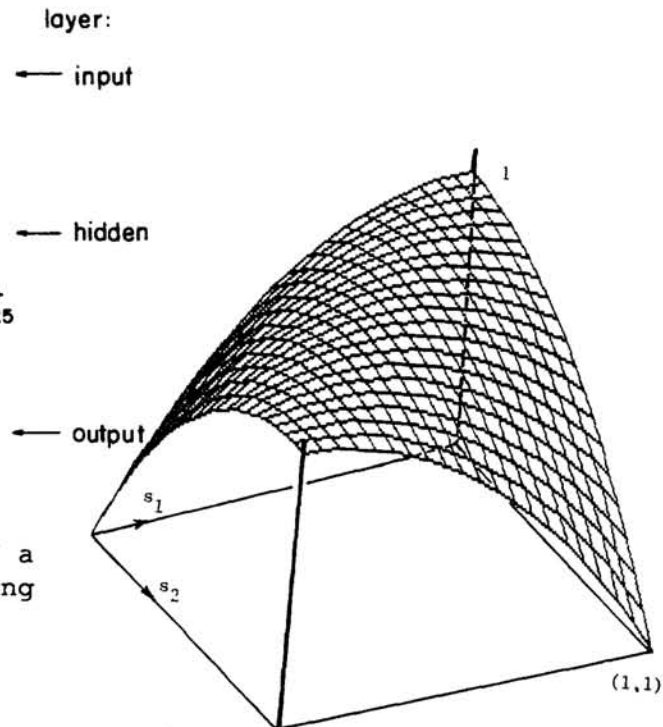

Figure 3. Response of the
elementary XOR APNN using an
exponential and trignometric
nonlinearity in the hidden
layer. Note that, at the
corners, the function is
equal to the XOR of the
coordinates.

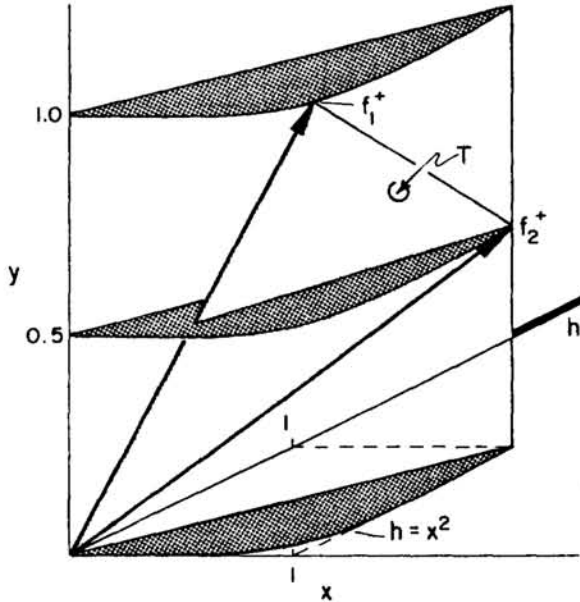

Figure 2. A geometrical
illustration of the use of an
$x^2$ nonlinearity to determine
the states of hidden neurons.

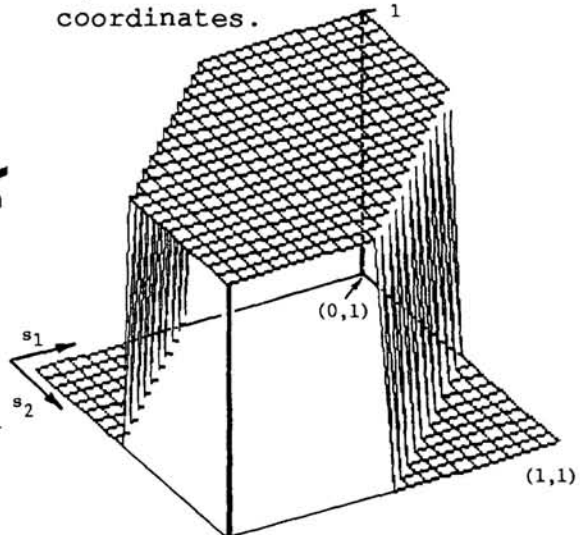

Figure 4. The generalization
of the XOR networks formed by
thresholding the function in
Fig.3 at 3/4. Different
hidden layer nonlinearities
result in different
generalizations.

iteration. If $s_1$ and $s_2$ denote the input clamped value, then $s_5(m+1)=t_{15}s_1 + t_{25}s_2 + t_{35}s_3 + t_{45}s_4 + t_{55}s_5(m)$ where $s_3=\exp(s_1+s_2)$ and $s_4=\cos[\pi(s_1 + s_2)/2]$ To reach steady state, we let m tend to infinity and solve for $s_5(\infty)$:

$$s_5(\infty) = \frac{1}{1-t_{55}}[t_{15}s_1 + t_{25}s_2 + t_{35}\exp(s_1+s_2) + t_{45}\cos\frac{\pi}{2}(s_1+s_2)] \tag{10}$$

A plot of $s_5(\infty)$ versus $(s_1,s_2)$ is shown in Figure 3. The plot goes through 1 and zero according to the XOR of the corner coordinates. Thresholding Figure 3 at 3/4 results in the generalization perspective plot shown in Figure 4.

To analyze the network's generalization when there are more than one output neuron, we use (5) of which (10) is a special case. If conditions are such that there is one step convergence, then generalization plots of the type in Figure 4 can be computed from one network iteration using (7).

## 7. NOTES

(a) There clearly exists a great amount of freedom in the choice of the nonlinearities in the hidden layer. Their effect on the network performance is currently not well understood. One can envision, however, choosing nonlinearities to enhance some network attribute such as interconnect reduction, classification region shaping (generalization) or convergence acceleration.

(b) There is a possibility that for a given set of hidden neuron nonlinearities, augmentation of the $\underline{F}_P$ matrix coincidentally will result in a matrix of deficent column rank, proper convergence is then not assured. It may also result in a poorly conditioned matrix, convergence will then be quite slow. A practical solution to these problems is to pad the hidden layer with additional neurons. As we have noted, this will improve the convergence rate.

(c) We have shown in section 4 that if an APNN has a single bipolar output neuron, the network converges in one step in the sense of (7). Visualize a layered APNN with a single output neuron. If there are a sufficiently large number of neurons in the hidden layer, then the input layer does not need to be connected to the output layer. Consider a second neural network identical to the first in the input and hidden layers except the hidden to output interconnects are different. Since the two networks are different only in the output interconnects, the two networks can be combined into a singlee network with two output neurons. The interconnects from the hidden layer to the output neurons are identical to those used in the single output neurons architectures. The new network will also converge in one step. This process can clearly be extended to an arbitrary number of output neurons.

## REFERENCES

1. R.J. Marks II, "A Class of Continuous Level Associative Memory Neural Nets," _Appl. Opt._, vol.26, no.10, p.2005, 1987.

2. K.F. Cheung *et. al.*, "Neural Net Associative Memories Based on Convex Set Projections," Proc. IEEE 1st International Conf. on Neural Networks, San Diego, 1987.

3. R.J. Marks II et. *al.*, "A Class of Continuous Level Neural Nets," Proc. 14th Congress of International Commission for Optics Conf., Quebec, Canada, 1987.

4. J.J. Hopfield, "Neural Networks and Physical Systems with Emergent Collective Computational Abilities," Proceedings Nat. Acad. of Sciences, USA, vol.79, p.2554, 1982.

5. J.J. Hopfield *et. al.*, "Neural Computation of Decisions in Optimization Problem," Biol. Cyber., vol.52, p.141, 1985.

6. D.W. Tank *et. al.*, "Simple Neurel Optimization Networks: an A/D Converter, Signal Decision Circuit and a Linear Programming Circuit," IEEE Trans. Cir. Sys., vol. CAS-33, p.533, 1986.

7. M. Takeda *et. al*, "Neural Networks for Computation: Number Representation and Programming Complexity," Appl. Opt., vol. 25, no. 18, p.3033, 1986.

8. S. Geman et. *al.*, "Stochastic Relaxation, Gibb's Distributions, and the Bayesian Restoration of Images," IEEE Trans. Pattern Recog. & Machine Intelligence., vol. PAMI-6, p.721, 1984.

9. S. Kirkpatrick *et. al.* ,"Optimization by Simulated Annealing," Science, vol. 220, no. 4598, p.671, 1983.

10. D.H. Ackley *et. al.*, "A Learning Algorithm for Boltzmann Machines," Cognitive Science, vol. 9, p.147, 1985.

11. K.F. Cheung *et. al.*, "Synchronous vs. Asynchronous Behaviour of Hopfield's CAM Neural Net," to appear in Applied Optics.

12. R.P. Lippmann, "An Introduction to Computing With Neural nets," IEEE ASSP Magazine, p.7, Apr 1987.

13. N. Farhat *et. al.*, "Optical Implementation of the Hopfield Model," Appl. Opt., vol. 24, pp.1469, 1985.

14. L.E. Atlas, "Auditory Coding in Higher Centers of the CNS," IEEE Eng. in Medicine and Biology Magazine, p.29, Jun 1987.

15. Y.S. Abu-Mostafa et. al., "Information Capacity of the Hopfield Model, " IEEE Trans. Inf. Theory, vol. IT-31, p.461, 1985.

16. R.J. McEliece *et. al.*,"The Capacity of the Hopfield Associative Memory, " IEEE Trans. Inf. Theory (submitted), 1986.

17. D.E. Rumelhart *et. al.*, **Parallel Distributed Processing**, vol. I & II, Bradford Books, Cambridge, MA, 1986.

18. D.E. Rumelhart *et. al.*, "Learning Representations by Back-Propagation Errors," Nature. vol. 323, no. 6088, p.533, 1986.

19. R.J. Marks II *et. al.*,"Alternating Projection Neural Networks," ISDL report #11587, Nov. 1987 (Submitted for publication).

20. D.C. Youla *et. al*, "Image Restoration by the Method of Convex Projections: Part I-Theory," IEEE Trans. Med. Imaging, vol. MI-1, p.81, 1982.

21. M.I. Sezan and H. Stark. "Image Restoration by the Method of Convex Projections: Part II-Applications and Numerical Results," IEEE Trans. Med. Imaging, vol. MI-1, p.95, 1985.

22. M. Minsky *et. al.*, **Perceptrons**, MIT Press, Cambridge, MA, 1969.

23. J. Sklansky *et. al.*, **Pattern Classifiers and Trainable Machines**, Springer-Verlag, New York, 1981.
